# A Dynamical Systems Approach for a Learnable Autonomous Robot

**Jun Tani and Naohiro Fukumura**
Sony Computer Science Laboratory Inc.
Takanawa Muse Building, 3-14-13 Higashi-gotanda, Shinagawa-ku,Tokyo, 141 JAPAN

## Abstract

This paper discusses how a robot can learn goal-directed navigation tasks using local sensory inputs. The emphasis is that such learning tasks could be formulated as an embedding problem of dynamical systems: desired trajectories in a task space should be embedded into an adequate sensory-based internal state space so that an unique mapping from the internal state space to the motor command could be established. The paper shows that a recurrent neural network suffices in self-organizing such an adequate internal state space from the temporal sensory input. In our experiments, using a real robot with a laser range sensor, the robot navigated robustly by achieving dynamical coherence with the environment. It was also shown that such coherence becomes structurally stable as the global attractor is self-organized in the coupling of the internal and the environmental dynamics.

## 1 Introduction

Conventionally, robot navigation problems have been formulated assuming a global view of the world. Given a detailed map of the workspace, described in a global coordinate system, the robot navigates to the specified goal by following this map. However, in situations where robots have to acquire navigational knowledge based on their own behaviors, it is important to describe the problems from the internal views of the robots.

[Kuipers 87], [Mataric 92] and others have developed an approach based on landmark detection. The robot acquires a graph representation of landmark types as a topological modeling of the environment through its exploratory travels using the local sensory inputs. In navigation, the robot can identify its topological position by anticipating the landmark types in the graph representation obtained. It is, however, considered that this navigation strategy might be susceptible to erroneous landmark-matching. If the robot is once lost by such a catastrophe, its recoverance of the positioning might be difficult. We need certain mechanisms by which the

robot can recover autonomously from such failures.

We study the above problems by using the dynamical systems approach, expecting that this approach would provide an effective representational and computational framework. The approach focuses on the fundamental dynamical structure that arises from coupling the internal and the environmental dynamics [Beer 95]. Here, the objective of learning is to adapt the internal dynamical function such that the resultant dynamical structure might generate the desired system behavior. The system's performance becomes structurally stable if the dynamical structure maintains a sufficiently large basin of attraction against possible perturbations.

We verify our claims through the implementation of our scheme on *YAMABICO* mobile robot equipped with a laser range sensor. The robot conducts navigational tasks under the following assumptions and conditions. (1) The robot cannot access its global position, but it navigates depending on its local sensory (range image) input. (2) There is no explicit landmarks accessible to the robot in the adopted workspace. (3) The robot learns tasks of *cyclic routing* by following guidance of a trainer. (4) The navigation should be robust enough against possible noise in the environment.

## 2   NAVIGATION ARCHITECTURE

The *YAMABICO* mobile robot [Yuta and Iijima 90] was used as an experimental platform. The robot can obtain range images by a range finder consisting of laser projectors and three CCD cameras. The ranges for 24 directions, covering a 160 degree arc in front of the robot, are measured every 150 milliseconds. In our formulation, maneuvering commands are generated as the output of a composite system consisting of two levels [Tani and Fukumura 94]. The control level generates a collision-free, smooth trajectory using the range image, while the navigation level directs the control level in a macroscopic sense, responding to the sequential branching that appears in the sensory flows. The control level is fixed; the navigation level, on the other hand, can be adapted through learning. Firstly, let us describe the control level. The robot can sense the forward range readings of the surrounding environment, given in robot-centered polar coordinates by $r_i$ $(1 \leq i \leq N)$. The angular range profile $R_i$ is obtained by smoothing the original range readings through applying an appropriate Gaussian filter. The maneuvering focus of the robot is the maximum (the angular direction of the largest range) in this range profile. The robot proceeds towards the maximum of the profile (an open space in the environment). The navigation level focuses on the topological changes in the range profile as the robot moves. As the robot moves through a given workspace, the profile gradually changes until another local peak appears when the robot reaches a branching point. At this moment of branching the navigation level decides whether to transfer the focus to the new local peak or to remain with the current one. It is noted that this branching could be quite undeterministic one if applied to rugged obstacle environment. The robot is likely to fail to detect branching points frequently in such environment.

The navigation level determines the branching by utilizing the range image obtained at branch points. Since the pertinent information in the range profile at a given moment is assumed to be only a small fraction of the total, we employ a vector quantization technique, known as the Kohonen network [Kohonen 82], so that the information in the profile may be compressed into specific lower-dimensional data. The Kohonen network employed here consists of an $l$-dimensional lattice with $m$ nodes along each dimension ($l=3$ and $m=6$ for the experiments with *YAMABICO*). The range image consisting of 24 values is input to the lattice, then the most

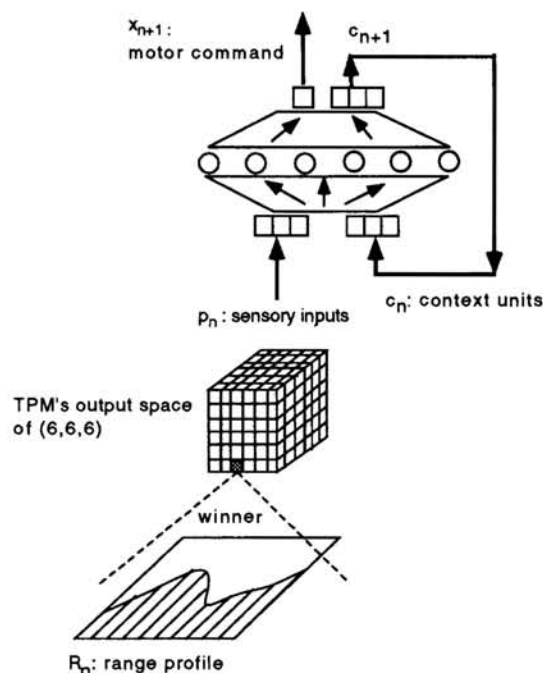

Figure 1: Neural architecture for skill-based learning.

highly activated unit in the lattice, the "winner" unit, is found. The address of the winner unit in the lattice denotes the output vector of the network. Therefore, the navigation level receives the sensory input compressed into three dimensional data. The next section will describe how the robot can generate right branching sequences upon receiving the compressed range image.

## 3 Formulation

### 3.1 Learning state-action map

The neural adaptation schemes are applied to the navigation level so that it can generate an adequate state-action map for a given task. Although some might consider that such map can be represented by using a layered feed-forward network with the inputs of the sensory image and the outputs of the motor command, this is not always true. The local sensory input does not always correspond uniquely to the true state of the robot (the sensory inputs could be the same for different robot positions). Therefore, there exists an ambiguity in determining the motor command solely from sensory inputs. This is a typical example of so-called non-Markovian problems which have been discussed by Lin and Mitchell [Lin and Mitchell 92]. In order to solve this ambiguity, a representation of contexts which are memories of past sensory sequences is required. For this purpose, a recurrent neural network (RNN) [Elman 90] was employed since its recurrent context states could represent the memory of past sequences. The employed neural architecture is shown in Figure. 1. The sensory input $p_n$ and the context units $c_n$ determine the appropriate motor command $x_{n+1}$. The motor command $x_n$ takes a binary value of 0 (staying at the current branch) or 1 (a transit to a new branch). The RNN learning of sensory-motor $(p_n, x_{n+1})$ sequences, sampled through the supervised training, can build the desired state-action map by self-organizing adequate internal representation in time.

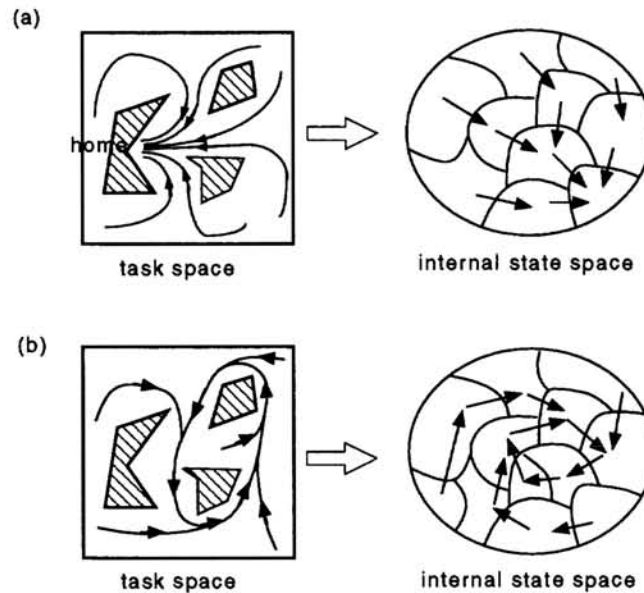

Figure 2: The desired trajectories in the task space and its mapping to the internal state space.

## 3.2 Embedding problem

The objective of the neural learning is to embed a task into certain global attractor dynamics which are generated from the coupling of the internal neural function and the environment. Figure 2 illustrates this idea. We define the internal state of the robot by the state of the RNN. The internal dynamics, which are coupled with the environmental dynamics through the sensory-motor loop, evolve as the robot travels in the task space. We assume that the desired vector field in the task space forms a global attractor, such as a fixed point for a homing task or limit cycling for a cyclic routing task. All that the robot has to do is to follow this vector flow by means of its internal state-action map. This requires a condition: the vector field in the internal state space should be self-organized as being topologically equivalent to that in the task space in order that the internal state determine the action (motor command) uniquely. This is the embedding problem from the task space to the internal state space, and RNN learning can attain this, using various training trajectories. This analysis conjectured that the trajectories in the task space can always converge into the desired one as long as the task is embedded into the global attractor in the internal state space.

## 4 Experiment

### 4.1 Task and training procedure

Figure 3 shows an example of the navigation task, (which is adopted for the physical experiment in a later section). The task is for the robot to repeat looping of a figure of '8' and '0' in sequence. The task is not trivial because at the branching position $A$ the robot has to decide whether to go '8' or '0' depending on its memory of the last sequence.

The robot learns this navigation task through supervision by a trainer. The trainer repeatedly guides the robot to the desired loop from a set of arbitrarily selected

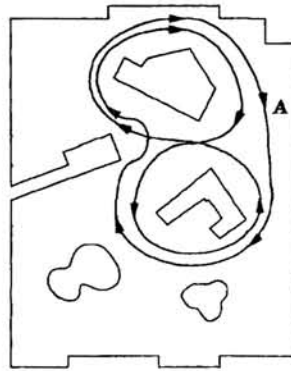

Figure 3: Cyclic routing task, in which *YAMABICO* has to trace a figure of eight followed by a single loop.

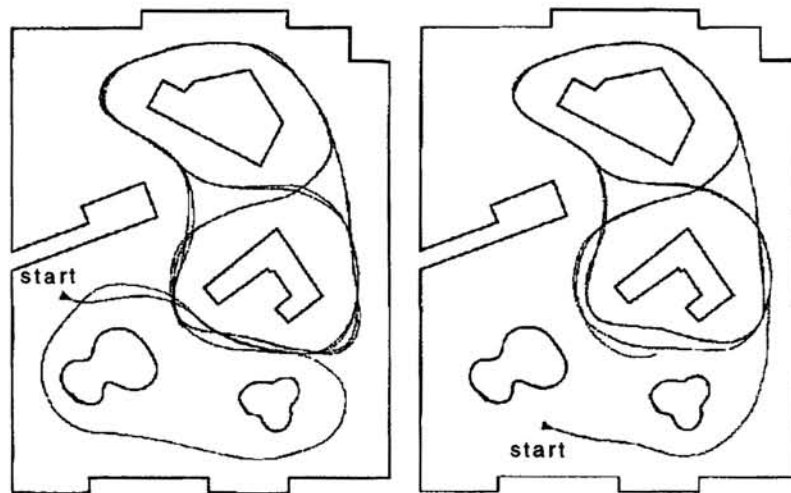

Figure 4: Trace of test travels for cyclic routing.

initial locations. (The training was conducted with starting the robot from 10 arbitrarily selected initial locations in the workspace.) In actual training, the robot moves by the navigation of the control level and stops at each branching point, where the branching direction is taught by the trainer. The sequence of range images and teaching branching commands at those bifurcation points are fed into the neural architecture as training data. The objective of training RNN is to find the optimal weight matrix that minimizes the mean square error of the training output (branching decision) sequences associating with sensory inputs (outputs of Kohonen network). The weight matrix can be obtained through an iterative calculation of back-propagation through time (BPTT) [Rumelhart *et al.* 86].

## 4.2 Results

After the training, we examined how the robot achieves the trained task. The robot was started from arbitrary initial positions for this test. Fig. 4 shows example test travels. The result showed that the robot always converged to the desired loop regardless of its starting position. The time required to converge, however, took a

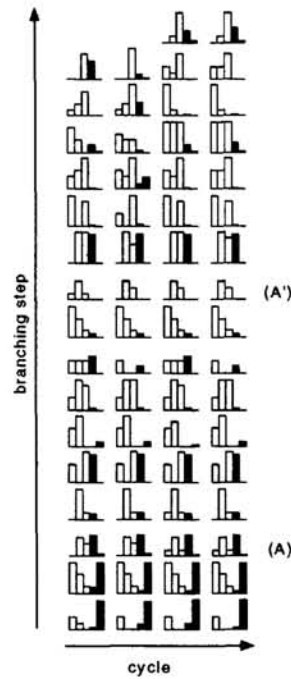

Figure 5: The sequence of activations in input and context units during the cycling travel.

certain period that depended on the case. The RNN initially could not function correctly because of the arbitrary initial setting of the context units. However, while the robot wandered around the workspace, the RNN became *situated* (recovered the context) as it encountered pre-learned sensory sequences. Thereafter, its navigation converged to the cycling loop.

Even after convergence, the robot could, by chance, leave the loop, under the influence of noise. However, the robot always came back to the loop after a while. These observations indicate that the robot learned the objective navigational task as embedded in a global attractor of limit cycling.

It is interesting to examine how the task is encoded in the internal dynamics of the RNN. We investigated the activation patterns of RNN after its convergence into the loop. The results are shown in Fig. 5. The input and context units at each branching point are shown as three white and two black bars, respectively. One cycle (the completion of two routes of '0' and '8') are aligned vertically as one column. The figure shows those of four continuous cycles. It can be seen that robot navigation is exposed to much noise; the sensing input vector becomes unstable at particular locations, and the number of branchings in one cycle is not constant (i.e. some branching points are undeterministic). The rows labeled as (A) and (A') are branches to the routes of '0' and '8', respectively. In this point, the sensory input receives noisy chattering of different patterns independent of (A) or (A'). The context units, on the other hand, is completely identifiable between (A) and (A'), which shows that the task sequence between two routes (a single loop and an eight) is rigidly encoded internally, even in a noisy environment. In further experiments in more rugged obstacle environments, we found that this sort of structural stability could not be always assured. When the undeterministicity in the branching exceeds a certain limit, the desired dynamical structure cannot be preserved.

## 5  Summary and Discussion

The navigation learning problem was formulated from the dynamical systems perspective. Our experimental results showed that the robot can learn the goal-directed navigation by embedding the desired task trajectories in the internal state space through the RNN training. It was also shown that the robot achieves the navigational tasks in terms of convergence of attractor dynamics which emerge in the coupling of the internal and the environmental dynamics. Since the dynamical coherence arisen in this coupling leads to the robust navigation of the robot, the intrinsic mechanism presented here is characterized by the term "autonomy".

Finally, it is interesting to study how robots can obtain analogical models of the environment rather than state-action maps for adapting to flexibly changed goals. We discuss such formulation based on the dynamical systems approach elsewhere [Tani 96].

## References

[Beer 95] R.D. Beer. A dynamical systems perspective on agent-environment interaction. *Artificial Intelligence*, Vol. 72, No. 1, pp.173–215, 1995.

[Elman 90] J.L. Elman. Finding structure in time. *Cognitive Science*, Vol. 14, pp.179–211, 1990.

[Kohonen 82] T. Kohonen. Self-Organized Formation of Topographically Correct Feature Maps. *Biological Cybernetics*, Vol. 43, pp.59–69, 1982.

[Kuipers 87] B. Kuipers. A Qualitative Approach to Robot Exploration and Map Learning. In *AAAI Workshop Spatial Reasoning and Multi-Sensor Fusion (Chicago)*, 1987.

[Lin and Mitchell 92] L.-J. Lin and T.M. Mitchell. Reinforcement learning with hidden states. In *Proc. of the Second Int. Conf. on Simulation of Adaptive Behavior*, pp. 271–280, 1992.

[Mataric 92] M. Mataric. Integration of Representation into Goal-driven Behavior-based Robot. *IEEE Trans. Robotics and Automation*, Vol. 8, pp.304–312, 1992.

[Rumelhart *et al.* 86] D.E. Rumelhart, G.E. Hinton, and R.J. Williams. Learning Internal Representations by Error Propagation. In *Parallel Distributed Processing*. MIT Press, 1986.

[Tani 96] J. Tani. Model-Based Learning for Mobile Robot Navigation from the Dynamical Systems Perspective. *IEEE Trans. System, Man and Cybernetics Part B, Special issue on robot learning*, Vol. 26, No. 3, 1996.

[Tani and Fukumura 94] J. Tani and N. Fukumura. Learning goal-directed sensory-based navigation of a mobile robot. *Neural Networks*, Vol. 7, No. 3, pp.553–563, 1994.

[Yuta and Iijima 90] S. Yuta and J. Iijima. State Information Panel for Inter-Processor Communication in an Autonomous Mobile Robot Controller. In *proc. of IROS'90*, 1990.
